# A Rational Analysis of Cognitive Control in a Speeded Discrimination Task

**Michael C. Mozer[+*], Michael D. Colagrosso[+*], David E. Huber[#*]**
[+] Department of Computer Science
[#] Department of Psychology
[*] Institute of Cognitive Science
University of Colorado
Boulder, CO 80309
{mozer,colagrom,dhuber}@colorado.edu

## Abstract

We are interested in the mechanisms by which individuals monitor and adjust their performance of simple cognitive tasks. We model a speeded discrimination task in which individuals are asked to classify a sequence of stimuli (Jones & Braver, 2001). Response conflict arises when one stimulus class is infrequent relative to another, resulting in more errors and slower reaction times for the infrequent class. How do control processes modulate behavior based on the relative class frequencies? We explain performance from a rational perspective that casts the goal of individuals as minimizing a cost that depends both on error rate and reaction time. With two additional assumptions of rationality—that class prior probabilities are accurately estimated and that inference is optimal subject to limitations on rate of information transmission—we obtain a good fit to overall RT and error data, as well as trial-by-trial variations in performance.

Consider the following scenario: While driving, you approach an intersection at which the traffic light has already turned yellow, signaling that it is about to turn red. You also notice that a car is approaching you rapidly from behind, with no indication of slowing. Should you stop or speed through the intersection? The decision is difficult due to the presence of two conflicting signals. Such *response conflict* can be produced in a psychological laboratory as well. For example, Stroop (1935) asked individuals to name the color of ink on which a word is printed. When the words are color names incongruous with the ink color—e.g., "blue" printed in red—reaction times are slower and error rates are higher. We are interested in the control mechanisms underlying performance of high-conflict tasks. Conflict requires individuals to monitor and adjust their behavior, possibly responding more slowly if errors are too frequent.

In this paper, we model a speeded discrimination paradigm in which individuals are asked to classify a sequence of stimuli (Jones & Braver, 2001). The stimuli are letters of the alphabet, **A**–**Z**, presented in rapid succession. In a *choice task*, individuals are asked to press one response key if the letter is an **X** or another response key for any letter other than **X** (as a shorthand, we will refer to non-**X** stimuli as **Y**). In a *go/no-go task*, individuals

are asked to press a response key when **X** is presented and to make no response otherwise. We address both tasks because they elicit slightly different decision-making behavior. In both tasks, Jones and Braver (2001) manipulated the relative frequency of the **X** and **Y** stimuli; the ratio of presentation frequency was either 17:83, 50:50, or 83:17. Response conflict arises when the two stimulus classes are unbalanced in frequency, resulting in more errors and slower reaction times. For example, when **X**'s are frequent but **Y** is presented, individuals are predisposed toward producing the **X** response, and this predisposition must be overcome by the perceptual evidence from the **Y**.

Jones and Braver (2001) also performed an fMRI study of this task and found that anterior cingulate cortex (ACC) becomes activated in situations involving response conflict. Specifically, when one stimulus occurs infrequently relative to the other, event-related fMRI response in the ACC is greater for the low frequency stimulus. Jones and Braver also extended a neural network model of Botvinick, Braver, Barch, Carter, and Cohen (2001) to account for human performance in the two discrimination tasks. The heart of the model is a mechanism that monitors conflict—the posited role of the ACC—and adjusts response biases accordingly. In this paper, we develop a parsimonious alternative account of the role of the ACC and of how control processes modulate behavior when response conflict arises.

# 1   A RATIONAL ANALYSIS

Our account is based on a *rational analysis* of human cognition, which views cognitive processes as being optimized with respect to certain task-related goals, and being adaptive to the structure of the environment (Anderson, 1990). We make three assumptions of rationality: (1) perceptual inference is optimal but is subject to rate limitations on information transmission, (2) response class prior probabilities are accurately estimated, and (3) the goal of individuals is to minimize a cost that depends both on error rate and reaction time.

The heart of our account is an existing probabilistic model that explains a variety of facilitation effects that arise from long-term repetition priming (Colagrosso, in preparation; Mozer, Colagrosso, & Huber, 2000), and more broadly, that addresses changes in the nature of information transmission in neocortex due to experience. We give a brief overview of this model; the details are not essential for the present work.

The model posits that neocortex can be characterized by a collection of information-processing *pathways*, and any act of cognition involves coordination among pathways. To model a simple discrimination task, we might suppose a *perceptual pathway* to map the visual input to a semantic representation, and a *response pathway* to map the semantic representation to a response. The choice and go/no-go tasks described earlier share a perceptual pathway, but require different response pathways. The model is framed in terms of probability theory: pathway inputs and outputs are random variables and microinference in a pathway is carried out by Bayesian belief revision.

To elaborate, consider a pathway whose input at time $t$ is a discrete random variable, denoted $X(t)$, which can assume values $1, 2, 3, \ldots n_x$ corresponding to alternative input states. Similarly, the output of the pathway at time $t$ is a discrete random variable, denoted $Y(t)$, which can assume values $1, 2, 3, \ldots n_y$. For example, the input to the perceptual pathway in the discrimination task is one of $n_x = 26$ visual patterns corresponding to the letters of the alphabet, and the output is one of $n_y = 26$ letter identities. (This model is highly abstract: the visual patterns are enumerated, but the actual pixel patterns are not explicitly represented in the model. Nonetheless, the similarity structure among inputs can be captured, but we skip a discussion of this issue because it is irrelevant for the current work.) To present a particular input alternative, $i$, to the model for $T$ time steps, we clamp $X(t) = i$ for $t = 1 \ldots T$. The model computes a probability distribution over $Y$ given $X$, i.e., $P(Y(t) \mid X(1) \ldots X(t))$.

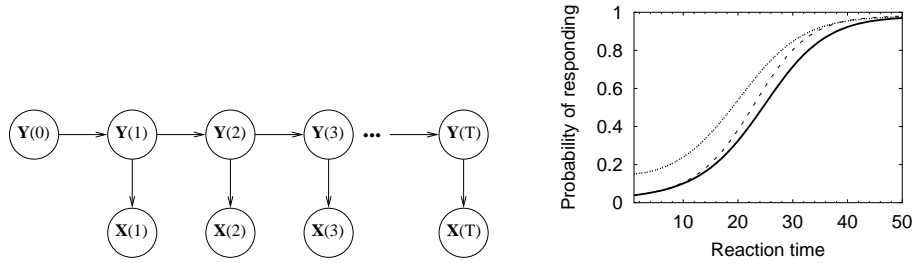

Figure 1: (left panel) basic pathway architecture—a hidden Markov model; (right panel) time course of inference in a pathway

A pathway is modeled as a dynamic Bayes network; the minimal version of the model used in the present simulations is simply a hidden Markov model, where the $X(t)$ are observations and the $Y(t)$ are inferred state (see Figure 1, left panel). (In typical usage, an HMM is presented with a sequence of distinct inputs, whereas we maintain the same input for many successive time steps. Further, in typical usage, an HMM transitions through a sequence of distinct hidden states, whereas we attempt to converge with increasing confidence on a single state. Thus, our model captures the time course of information processing for a single event.)

To compute $P(Y(t) \mid X(1) \ldots X(t))$, three probability distributions must be specified: (1) $P(Y(t) \mid Y(t-1))$, which characterizes how the pathway output evolves over time, (2) $P(X(t) \mid Y(t))$, which characterizes the *strength of association* between inputs and outputs, and (3) $P(Y(0))$, the *prior* distribution over outputs in the absence of any information about the input. The particular values hypothesized for these three distributions embody the knowledge of the model—like weights in a neural networks—and give rise to predictions from the model.

To give a sense of how the Mozer et al. (2000) model operates, the right panel of Figure 1 depicts the time course of inference in a single pathway which has 26 input and output alternatives, with one-to-one associations. The solid line in the Figure shows, as a function of time $t$, $P(Y(t) = 1 \mid X(1) = 1 \ldots X(t) = 1)$, i.e., the probability that a given input will produce its target output. Due to limited association strengths, perceptual evidence must accumulate over many iterations in order for the target to be produced with high probability. The densely dashed line shows the same target probability when the target prior is increased, and the sparsely dashed line shows the target probability when the association strength to the target is increased. Increasing either the prior or the association strength causes the speed-accuracy curve to shift to the left. In our previous work, we proposed a mechanism by which priors and association strengths are altered through experience.

## 1.1 Model Details

The simulations we report in this paper utilize two pathways in cascade. A perceptual pathway maps visual patterns (26 alternatives) to a letter-identity representation (26 alternatives), and a response pathway maps the letter identity to a response. For the choice task, the response pathway has two outputs, corresponding to the two response keys; for the go/no-go task, the response pathway also has two outputs, which are interpreted as "go" and "no go." The interconnection between the pathways is achieved by copying the output of the perceptual pathway, $Y^p(t)$, to the input of the response pathway, $X^r(t)$, at each time. The free parameters of the model are mostly task and experience related. Nonetheless, in the current simulations we used the same parameter values as Mozer et al. (2000), with one exception: Because the speeded perceptual discrimination task studied here is quite unlike

the tasks studied by Mozer et al., we allowed ourselves to vary the association-strength parameter in the response pathway. This parameter has only a quantitative, not qualitative, influence on predictions of the model.

In our simulations, we also use the priming mechanism proposed by Mozer et al. (2000), which we briefly describe. The priors for a pathway are internally represented in a nonnormalized form: the nonnormalized prior for alternative $i$ is $p_i$, and the normalized prior is $P(Y(0) = i) = p_i / \sum_j p_j$. On each trial, the priming mechanism increases the nonnormalized prior of alternative $i$ in proportion to its asymptotic activity at final time $T$, and and all priors undergo exponential decay: $\Delta p_i = \gamma P(Y(T) = i | X) - \epsilon p_i$, where $\gamma$ is the strength of priming, and $\epsilon$ is the decay rate. (The Mozer et al. model also performs priming in the association strengths by a similar rule, which is included in the present simulation although it has a negligible effect on the results here.)

This priming mechanism yields priors on average that match the presentation probabilities in the task, e.g., .17 and .83 for the two responses in the 17:83 condition of the Jones and Braver experiment. Consequently, when we report results for overall error rate and reaction time in a condition, we make the assumption of rationality that the model's priors correspond to the true priors of the environment. Although the model yields the same result when the priming mechanism is used on a trial-by-trial basis to adjust the priors, the explicit assumption of rationality avoids any confusion about the factors responsible for the model's performance. We use the priming mechanism on a trial-by-trial basis to account for performance conditional on recent trial history, as explained later.

## 1.2 Control Processes and the Speed-Accuracy Trade Off

The response pathway of the model produces a speed-accuracy performance function much like that in Figure 1b. This function characterizes the operation of the pathway, but it does not address the control issue of when in time to initiate a response. A control mechanism might simply choose a threshold in accuracy or in reaction time, but we hypothesize a more general, rational approach in which a *response cost* is computed, and control mechanisms initiate a response at the point in time when a minimum in cost is attained.

When stimulus $\mathbf{S}$ is presented and the correct response is $\mathbf{R}$, we posit a cost of responding at time $t$ following stimulus onset:

$$c(t \mid \mathbf{S}, \mathbf{R}) = P(Y^r(t) \neq \mathbf{R} \mid \mathbf{S}) + \kappa_{\mathbf{R}} t. \tag{1}$$

This cost involves two terms—the error rate and the reaction time—which are summed, with a weighting factor, $\kappa_{\mathbf{R}}$, that determines the relative importance of the two terms. We assume that $\kappa$ is dependent on task instructions: if individuals are told to make no errors, $\kappa$ should be small to emphasize the error rate; if individuals are told to respond quickly and not concern themselves with occasional errors, $\kappa$ should be large to emphasize the reaction time.

The cost $c(t \mid \mathbf{S}, \mathbf{R})$ cannot be computed without knowing the correct response $\mathbf{R}$. Nonetheless, the control mechanism could still compute an *expected* cost over the $n_y^r$ alternative responses based on the model's current estimate of the likelihood of each:

$$E[c(t \mid \mathbf{S}, \mathbf{R})] = \sum_{i=1}^{n_y^r} P(Y^r(t) = i \mid \mathbf{S}) c(t \mid \mathbf{S}, i) \tag{2}$$

It is this expected cost that is minimized to determine the appropriate point in time at which to respond. We index $\kappa_{\mathbf{R}}$ by the response $\mathbf{R}$ because it is not sensible to assign a time cost to a "no go" response, where no response is produced. Consequently, $\kappa_{no-go} = 0$; for the "go" response and for the two responses in the choice task, we searched for the parameter that best fit the data, yielding $\kappa = .05$.

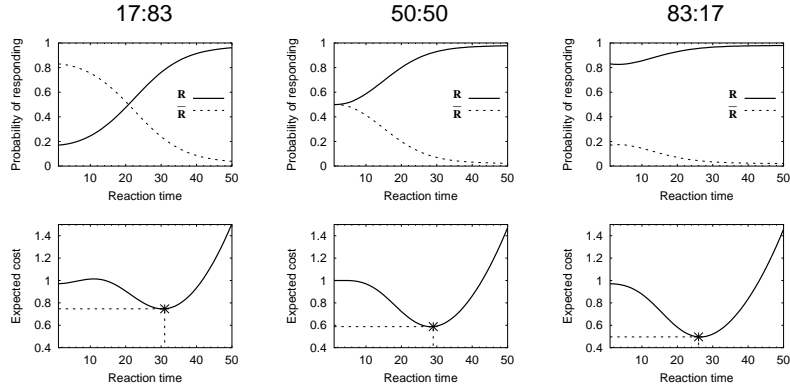

Figure 2: (upper row) Output of response pathway when stimulus **S**, associated with response **R**, is presented, and relative frequency of **R** and the alternative response, **R̄**, is 17:83, 50:50, and 83.17. (lower row) Expected cost of responding (Eqn. 2).

## 2 RESULTS

Figure 2 illustrates the model's performance on the choice task when presented with a stimulus, **S**, associated with a response, **R**, and the relative frequency of **R** and the alternative response, **R̄**, is 17:83, 50:50, or 83:17 (left, center, and right columns, respectively). Each graph in the top row plots the probability of **R** and **R̄** against time. Although **R** wins out asymptotically in all three conditions, it must overcome the effect of its low prior in the 17:83 condition. Each graph in the bottom row plots the expected cost, $c(t)$, over time. In the early part of the cost function, error rate dominates the cost, and in the late part, reaction time dominates. In fact, at long times, the error rate is essentially 0, and the cost grows linearly with reaction time. Our rational analysis suggests that a response should be initiated at the global minimum—indicated by asterisks in the figure—implying that both the reaction time and error rate will decrease as the response prior is increased.

Figure 3 presents human and simulation data for the choice task. The data consist of mean reaction time and accuracy for the two target responses, $\mathbf{R}_1$ and $\mathbf{R}_2$, for the three conditions corresponding to different $\mathbf{R}_1$:$\mathbf{R}_2$ presentation ratios. Figure 4 presents human and simulation data for the go/no-go task. Note that reaction times are shown only for the "go" trials, because no response is required for the "no go" trials. For both tasks, the model provides an extremely good fit to the human data. The qualities of the model giving rise to the fit can be inferred by inspection of Figure 2—namely, accuracy is higher and reaction times are faster when a response is expected.

Figure 5 reveals how the recent history of experimental trials influences reaction time and error rate in the choice task. The trial *context* along the x-axis is coded as $v_4 v_3 v_2 v_1$, where $v_i$ specifies that trial $n-i$ required the same ("S") or different ("D") response as trial $n-i+1$. For example, if the current trial required response **X**, and the four trials leading up to the current trial were—in forward temporal order—**Y**, **Y**, **Y**, and **X**, the current trial's context would be coded as "SSDS." The correlation coefficient between human and simulation data is .960 for reaction time and .953 for error rate.

The model fits the human data extremely well. The simple priming mechanism proposed previously by Mozer et al. (2000), which aims to adapt the model's priors rapidly to the statistics of the environment, is responsible: On a coarse time scale, the mechanism produces priors in the model that match priors in the environment. On a fine time scale, changes to and decay of the priors results in a strong effect of recent trial history, consistent with the human data: The graphs in Figure 5 show that the fastest and most accurate trials

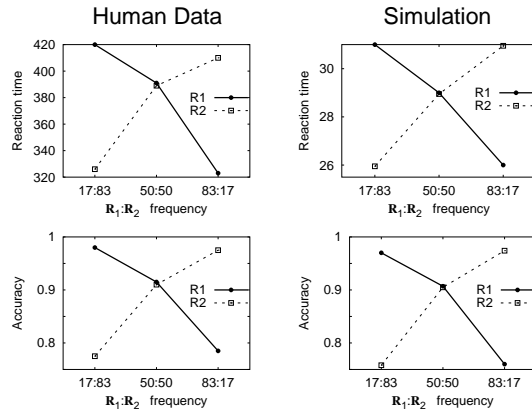

Figure 3: Human data (left column) and simulation results (right column) for the choice task. Human data from Jones and Braver (2001). The upper and lower rows show mean reaction time and accuracy, respectively, for the two responses ($R_1$ and $R_2$) in the three conditions corresponding to different $R_1$:$R_2$ frequencies.

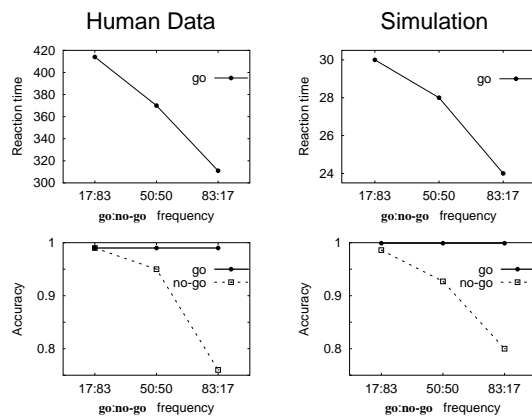

Figure 4: Human data (left column) and simulation results (right column) for the go/no-go task. Human data from Jones and Braver (2001). The upper and lower rows show mean reaction time and accuracy, respectively, for the two responses (**go** and **no-go**) in the three conditions corresponding to different **go**:**no-go** presentation frequencies.

are clearly those in which the previous two trials required the same response as the current trial (the leftmost four contexts in each graph). The fit to the data is all the more impressive given that Mozer et al. priming mechanism was used to model perceptual priming, and here the same mechanism is used to model response priming.

## 3 DISCUSSION

We introduced a probabilistic model based on three principles of rationality: (1) perceptual inference is optimal but is subject to rate limitations on information transmission, (2) response class prior probabilities are accurately estimated, and (3) the goal of individuals is to minimize a cost that depends both on error rate and reaction time. The model provides a parsimonious account of the detailed pattern of human data from two speeded discrimination tasks. The heart of the model was proposed previously by Mozer, Colagrosso, and Huber (2000), and in the present work we fit experimental data with only two free parameters, one relating to the rate of information flow, and the other specifying the relative cost of speed and errors. The simplicity and elegance of the model arises from having adopted the rational perspective, which imposes strong constraints on the model and removes arbitrary choices and degrees of freedom that are often present in psychological models.

Jones and Braver (2001) proposed a neural network model to address response conflict in a speeded discrimination task. Their model produces an excellent fit to the data too, but

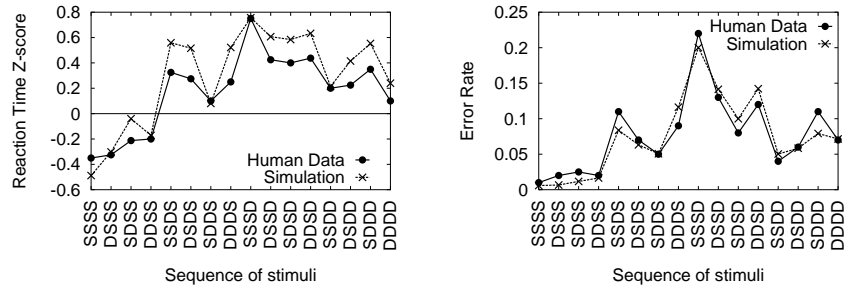

Figure 5: Reaction time (left curve) and accuracy (right curve) data for humans (solid line) and model (dashed line), contingent on the recent history of experimental trials.

involves significantly more machinery, free parameters, and ad hoc assumptions. In brief, their model is an associative net mapping activity from stimulus units to response units. When response units $R_1$ and $R_2$ both receive significant activation, noise in the system can push the inappropriate response unit over threshold. When this conflict situation is detected, a control mechanism acts to lower the baseline activity of response units, requiring them to build up more evidence before responding and thereby reducing the likelihood of noise determining the response. Their model includes a priming mechanism to facilitate repetition of responses, much as we have in our model. However, their model also includes a secondary priming mechanism to facilitate *alternation* of responses, which our model does not require. Both models address additional data; for example, a variant of their model predicts a neurophysiological marker of conflict called error-related negativity (Yeung, Botvinick, & Cohen, 2000).

Jones and Braver posit that the role of the ACC is conflict detection. Our account makes an alternative proposal—that *ACC activity reflects the expected cost of decision making*. Both hypotheses are consistent with the fMRI data indicating that the ACC produces a greater response for a low frequency stimulus. We are presently considering further experiments to distinguish these contrasting hypotheses.

## Acknowledgments

We thank Andrew Jones and Todd Braver for generously providing their raw data and for helpful discussions leading to this research. This research was supported by Grant 97–18 from the McDonnell-Pew Program in Cognitive Neuroscience, NSF award IBN–9873492, and NIH/IFOPAL R01 MH61549–01A1.

## References

Botvinick, M. M., Braver, T. S., Barch, D. M., Carter, C. S., & Cohen, J. D. (2001). Evaluating the demand for control: anterior cingulate cortex and conflict monitoring. Submitted for publication.

Colagrosso, M. (in preparation). A Bayesian cognitive architecture for analyzing information transmission in neocortex. Ph.D. Dissertation in preparation.

Jones, A. D., & Braver, T. S. (2001). Sequential modulations in control: Conflict monitoring and the anterior cingulate cortex. Submitted for publication.

Mozer, M. C., Colagrosso, M. D., & Huber, D. H. (2000). A Bayesian Cognitive Architecture for Interpreting Long-Term Priming Phenomena. Presentation at the 41st Annual Meeting of the Psychonomic Society, New Orleans, LA, November 2000.

Yeung, N., Botvinick, M. M., & Cohen, J. D. (2000). The neural basis of error detection: Conflict monitoring and the error-related negativity. Submitted for publication.
